# Learning Taxonomies by Dependence Maximization

**Matthew B. Blaschko**     **Arthur Gretton**
Max Planck Institute for Biological Cybernetics
Spemannstr. 38
72076 Tübingen, Germany
`{blaschko,arthur}@tuebingen.mpg.de`

## Abstract

We introduce a family of unsupervised algorithms, *numerical taxonomy clustering*, to simultaneously cluster data, and to learn a taxonomy that encodes the relationship between the clusters. The algorithms work by maximizing the dependence between the taxonomy and the original data. The resulting taxonomy is a more informative visualization of complex data than simple clustering; in addition, taking into account the relations between different clusters is shown to substantially improve the quality of the clustering, when compared with state-of-the-art algorithms in the literature (both spectral clustering and a previous dependence maximization approach). We demonstrate our algorithm on image and text data.

## 1   Introduction

We address the problem of finding taxonomies in data: that is, to cluster the data, and to specify in a systematic way how the clusters relate. This problem is widely encountered in biology, when grouping different species; and in computer science, when summarizing and searching over documents and images. One of the simpler methods that has been used extensively is agglomerative clustering [18]. One specifies a distance metric and a linkage function that encodes the cost of merging two clusters, and the algorithm greedily agglomerates clusters, forming a hierarchy until at last the final two clusters are merged into the tree root. A related alternate approach is divisive clustering, in which clusters are split at each level, beginning with a partition of all the data, e.g. [19]. Unfortunately, this is also a greedy technique and we generally have no approximation guarantees. More recently, hierarchical topic models [7, 23] have been proposed to model the hierarchical cluster structure of data. These models often rely on the data being representable by multinomial distributions over bags of words, making them suitable for many problems, but their application to arbitrarily structured data is in no way straightforward. Inference in these models often relies on sampling techniques that can affect their practical computational efficiency.

On the other hand, many kinds of data can be easily compared using a kernel function, which encodes the measure of similarity between objects based on their features. Spectral clustering algorithms represent one important subset of clustering techniques based on kernels [24, 21]: the spectrum of an appropriately normalized similarity matrix is used as a relaxed solution to a partition problem. Spectral techniques have the advantage of capturing global cluster structure of the data, but generally do not give a global solution to the problem of discovering taxonomic structure.

In the present work, we propose a novel unsupervised clustering algorithm, *numerical taxonomy clustering*, which both clusters the data and learns a taxonomy relating the clusters. Our method works by maximizing a kernel measure of dependence between the observed data, and a product of the partition matrix that defines the clusters with a structure matrix that defines the relationship between individual clusters. This leads to a constrained maximization problem that is in general NP hard, but that can be approximated very efficiently using results in spectral clustering and numerical

taxonomy (the latter field addresses the problem fitting taxonomies to pairwise distance data [1, 2, 4, 8, 11, 15, 25], and contains techniques that allow us to efficiently fit a tree structure to our data with tight approximation guarantees). Aside from its simplicity and computational efficiency, our method has two important advantages over previous clustering approaches. First, it represents a more informative visualization of the data than simple clustering, since the relationship between the clusters is also represented. Second, we find the clustering performance is improved over methods that do not take cluster structure into account, and over methods that impose a cluster distance structure rather than learning it.

Several objectives that have been used for clustering are related to the objective employed here. Bach and Jordan [3] proposed a modified spectral clustering objective that they then maximize either with respect to the kernel parameters or the data partition. Christianini et al. [10] proposed a normalized inner product between a kernel matrix and a matrix constructed from the labels, which can be used to learn kernel parameters. The objective we use here is also a normalized inner product between a similarity matrix and a matrix constructed from the partition, but importantly, we include a structure matrix that represents the relationship between clusters. Our work is most closely related to that of Song et al. [22], who used an objective that includes a fixed structure matrix and an objective based on the Hilbert-Schmidt Independence Criterion. Their objective is not normalized, however, and they do not maximize with respect to the structure matrix.

The paper is organized as follows. In Section 2, we introduce a family of dependence measures with which one can interpret the objective function of the clustering approach. The dependence maximization objective is presented in Section 3, and its relation to classical spectral clustering algorithms is explained in Section 3.1. Important results for the optimization of the objective are presented in Sections 3.2 and 3.3. The problem of numerical taxonomy and its relation to the proposed objective function is presented in Section 4, as well as the numerical taxonomy clustering algorithm. Experimental results are given in Section 5.

## 2   Hilbert-Schmidt Independence Criterion

In this section, we give a brief introduction to the Hilbert-Schmidt Independence Criterion (HSIC), which is a measure of the strength of dependence between two variables (in our case, following [22], these are the data before and after clustering). We begin with some basic terminology in kernel methods. Let $\mathcal{F}$ be a reproducing kernel Hilbert space of functions from $\mathcal{X}$ to $\mathbb{R}$, where $\mathcal{X}$ is a separable metric space (our input domain). To each point $x \in \mathcal{X}$, there corresponds an element $\phi(x) \in \mathcal{F}$ (we call $\phi$ the *feature map*) such that $\langle \phi(x), \phi(x') \rangle_{\mathcal{F}} = k(x, x')$, where $k : \mathcal{X} \times \mathcal{X} \to \mathbb{R}$ is a unique positive definite kernel. We also define a second RKHS $\mathcal{G}$ with respect to the separable metric space $\mathcal{Y}$, with feature map $\psi$ and kernel $\langle \psi(y), \psi(y') \rangle_{\mathcal{G}} = l(y, y')$.

Let $(X, Y)$ be random variables on $\mathcal{X} \times \mathcal{Y}$ with joint distribution $\Pr_{X,Y}$, and associated marginals $\Pr_X$ and $\Pr_Y$. Then following [5, 12], the covariance operator $C_{xy} : \mathcal{G} \to \mathcal{F}$ is defined such that for all $f \in \mathcal{F}$ and $g \in \mathcal{G}$,

$$\langle f, C_{xy} g \rangle_{\mathcal{F}} = \mathbf{E}_{\mathsf{x},\mathsf{y}} \left( [f(\mathsf{x}) - \mathbf{E}_{\mathsf{x}}(f(\mathsf{x}))] \, [g(\mathsf{y}) - \mathbf{E}_{\mathsf{y}}(g(\mathsf{y}))] \right).$$

A measure of dependence is then the Hilbert-Schmidt norm of this operator (the sum of the squared singular values), $\|C_{xy}\|_{\mathrm{HS}}^2$. For characteristic kernels [13], this is zero if and only if $X$ and $Y$ are independent. It is shown in [13] that the Gaussian and Laplace kernels are characteristic on $\mathbb{R}^d$. Given a sample of size $n$ from $\Pr_{X,Y}$, the Hilbert-Schmidt Independence Criterion (HSIC) is defined by [14] to be a (slightly biased) empirical estimate of $\|C_{xy}\|_{\mathrm{HS}}^2$,

$$\mathrm{HSIC} := \mathrm{Tr} \left[ H_n K H_n L \right], \quad \text{where} \quad H_n = I - \frac{1}{n} 1_n 1_n^T,$$

$1_n$ is the $n \times 1$ vector of ones, $K$ is the Gram matrix for samples from $\Pr_X$ with $(i, j)$th entry $k(x_i, x_j)$, and L is the Gram matrix with kernel $l(y_i, y_j)$.

## 3   Dependence Maximization

We now specify how the dependence criteria introduced in the previous section can be used in clustering. We represent our data via an $n \times n$ Gram matrix $M \succeq \mathbf{0}$: in the simplest case, this

is the centered kernel matrix ($M = H_n K H_n$), but we also consider a Gram matrix corresponding to normalized cuts clustering (see Section 3.1). Following [22], we define our output Gram matrix to be $L = \Pi Y \Pi^T$, where $\Pi$ is an $n \times k$ partition matrix, $k$ is the number of clusters, and $Y$ is a positive definite matrix that encodes the relationship between clusters (e.g. a taxonomic structure). Our clustering quality is measured according to

$$\frac{\operatorname{Tr}\left[M H_n \Pi Y \Pi^T H_n\right]}{\sqrt{\operatorname{Tr}\left[\Pi Y \Pi^T H_n \Pi Y \Pi^T H_n\right]}}. \tag{1}$$

In terms of the covariance operators introduced earlier, we are optimizing HSIC, this being an empirical estimate of $\|C_{xy}\|_{\mathrm{HS}}^2$, while normalizing by the empirical estimate of $\|C_{yy}\|_{\mathrm{HS}}^2$ (we need not normalize by $\|C_{xx}\|_{\mathrm{HS}}^2$, since it is constant). This criterion is very similar to the criterion introduced for use in kernel target alignment [10], the difference being the addition of centering matrices, $H_n$, as required by definition of the covariance. We remark that the normalizing term $\left\|H_n \Pi Y \Pi^T H_n\right\|_{\mathrm{HS}}$ was not needed in the structured clustering objective of [22]. This is because Song et al. were interested only in solving for the partition matrix, $\Pi$, whereas we also wish to solve for $Y$: without normalization, the objective can always be improved by scaling $Y$ arbitrarily. In the remainder of this section, we address the maximization of Equation (1) under various simplifying assumptions: these results will then be used in our main algorithm in Section 4.

## 3.1 Relation to Spectral Clustering

Maximizing Equation (1) is quite difficult given that the entries of $\Pi$ can only take on values in $\{0, 1\}$, and that the row sums have to be equal to 1. In order to more efficiently solve this difficult combinatorial problem, we make use of a spectral relaxation. Consider the case that $\Pi$ is a column vector and $Y$ is the identity matrix. Equation (1) becomes

$$\max_{\Pi} \frac{\operatorname{Tr}\left[M H_n \Pi \Pi^T H_n\right]}{\sqrt{\operatorname{Tr}\left[\Pi \Pi^T H_n \Pi \Pi^T H_n\right]}} = \max_{\Pi} \frac{\Pi^T H_n M H_n \Pi}{\Pi^T H_n \Pi} \tag{2}$$

Setting the derivative with respect to $\Pi$ to zero and rearranging, we obtain

$$H_n M H_n \Pi = \frac{\Pi^T H_n M H_n \Pi}{\Pi^T H_n \Pi} H_n \Pi. \tag{3}$$

Using the normalization $\Pi^T H_n \Pi = 1$, we obtain the generalized eigenvalue problem

$$H_n M H_n \Pi_i = \rho_i H_n \Pi_i, \quad \text{or equivalently} \quad H_n M H_n \Pi_i = \rho_i \Pi_i. \tag{4}$$

For $\Pi \in \{0, 1\}^{n \times k}$ where $k > 1$, we can recover $\Pi$ by extracting the $k$ eigenvectors associated with the largest eigenvalues. As discussed in [24, 21], the relaxed solution will contain an arbitrary rotation which can be recovered using a reclustering step.

If we choose $M = D^{-\frac{1}{2}} A D^{-\frac{1}{2}}$ where $A$ is a similarity matrix, and $D$ is the diagonal matrix such that $D_{ii} = \sum_j A_{ij}$, we can recover a centered version of the spectral clustering of [21]. In fact, we wish to ignore the eigenvector with constant entries [24], so the centering matrix $H_n$ does not alter the clustering solution.

## 3.2 Solving for Optimal $Y \succeq 0$ Given $\Pi$

We now address the subproblem of solving for the optimal structure matrix, $Y$, subject only to positive semi-definiteness, for any $\Pi$. We note that the maximization of Equation (1) is equivalent to the constrained optimization problem

$$\max_{Y} \quad \operatorname{Tr}\left[M H_n \Pi Y \Pi^T H_n\right], \quad \text{s.t.} \quad \operatorname{Tr}\left[\Pi Y \Pi^T H_n \Pi Y \Pi^T H_n\right] = 1 \tag{5}$$

We write the Lagrangian

$$\mathcal{L}(Y, \nu) = \operatorname{Tr}\left[M H_n \Pi Y \Pi^T H_n\right] + \nu\left(1 - \operatorname{Tr}\left[\Pi Y \Pi^T H_n \Pi Y \Pi^T H_n\right]\right), \tag{6}$$

take the derivative with respect to $Y$, and set to zero, to obtain

$$\frac{\partial \mathcal{L}}{\partial Y} = \Pi^T H_n M H_n \Pi - 2\nu\left(\Pi^T H_n \Pi Y \Pi^T H_n \Pi\right) = 0 \tag{7}$$

which together with the constraint in Equation (5) yields

$$Y^* = \frac{\left(\Pi^T H_n \Pi\right)^\dagger \Pi^T H_n M H_n \Pi \left(\Pi^T H_n \Pi\right)^\dagger}{\sqrt{\text{Tr}\left[\Pi^T H_n M H_n \Pi \left(\Pi^T H_n \Pi\right)^\dagger \Pi^T H_n M H_n \Pi \left(\Pi^T H_n \Pi\right)^\dagger\right]}}, \tag{8}$$

where $\dagger$ indicates the Moore-Penrose generalized inverse [17, p. 421].

Because $\left(\Pi^T H_n \Pi\right)^\dagger \Pi^T H_n = H_k \left(\Pi^T \Pi\right)^{-1} \Pi^T H_n$ (see [6, 20]), we note that Equation (8) computes a normalized set kernel between the elements in each cluster. Up to a constant normalization factor, $Y^*$ is equivalent to $H_k \tilde{Y}^* H_k$ where

$$\tilde{Y}_{ij}^* = \frac{1}{N_i N_j} \sum_{\iota \in C_i} \sum_{\kappa \in C_j} \tilde{M}_{\iota \kappa}, \tag{9}$$

$N_i$ is the number of elements in cluster $i$, $C_i$ is the set of indices of samples assigned to cluster $i$, and $\tilde{M} = H_n M H_n$. This is a standard set kernel as defined in [16].

### 3.3   Solving for $\Pi$ with the Optimal $Y \succeq 0$

As we have solved for $Y^*$ in closed form in Equation (8), we can plug this result into Equation (1) to obtain a formulation of the problem of optimizing $\Pi^*$ that does not require a simultaneous optimization over $Y$. Under these conditions, Equation (1) is equivalent to

$$\max_\Pi \sqrt{\text{Tr}\left[\Pi^T H_n M H_n \Pi \left(\Pi^T \Pi\right)^{-1} \Pi^T H_n M H_n \Pi \left(\Pi^T \Pi\right)^{-1}\right]}. \tag{10}$$

By evaluating the first order conditions on Equation (10), we can see that the relaxed solution, $\Pi^*$, to Equation (10) must lie in the principal subspace of $H_n M H_n$.[1] Therefore, for the problem of simultaneously optimizing the structure matrix, $Y \succeq \mathbf{0}$, and the partition matrix, one can use the same spectral relaxation as in Equation (4), and use the resulting partition matrix to solve for the optimal assignment for $Y$ using Equation (8). This indicates that the optimal partition of the data is the same for $Y$ given by Equation (8) and for $Y = I$. We show in the next section how we can add additional constraints on $Y$ to not only aid in interpretation, but to actually improve the optimal clustering.

## 4   Numerical Taxonomy

In this section, we consolidate the results developed in Section 3 and introduce the numerical taxonomy clustering algorithm. The algorithm allows us to simultaneously cluster data and learn a tree structure that relates the clusters. The tree structure imposes constraints on the solution, which in turn affect the data partition selected by the clustering algorithm. The data are only assumed to be well represented by some taxonomy, but not any particular topology or structure.

In Section 3 we introduced techniques for solving for $Y$ and $\Pi$ that depend only on $Y$ being constrained to be positive semi-definite. In the interests of interpretability, as well as the ability to influence clustering solutions by prior knowledge, we wish to explore the problem where additional constraints are imposed on the structure of $Y$. In particular, we consider the case that $Y$ is constrained to be generated by a tree metric. By this, we mean that the distance between any two clusters is consistent with the path length along some fixed tree whose leaves are identified with the clusters. For any positive semi-definite matrix $Y$, we can compute the distance matrix, $D$, given by the norm implied by the inner product that computes $Y$, by assigning $D_{ij} = \sqrt{Y_{ii} + Y_{jj} - 2Y_{ij}}$. It is sufficient, then, to reformulate the optimization problem given in Equation (1) to add the following constraints that characterize distances generated by a tree metric

$$D_{ab} + D_{cd} \leq \max\left(D_{ac} + D_{bd}, D_{ad} + D_{bc}\right) \quad \forall a, b, c, d, \tag{11}$$

where $D$ is the distance matrix generated from $Y$. The constraints in Equation (11) are known as the 4-point condition, and were proven in [8] to be necessary and sufficient for $D$ to be a tree metric.

Optimization problems incorporating these constraints are combinatorial and generally difficult to solve. The problem of *numerical taxonomy*, or fitting additive trees, is as follows: given a fixed distance matrix, $D$, that fulfills metric constraints, find the solution to

$$\min_{D_T} \|D - D_T\|^2 \tag{12}$$

with respect to some norm (e.g. $L^1$, $L^2$, or $L^\infty$), where $D_T$ is subject to the 4-point condition. While numerical taxonomy is in general NP hard, a great variety of approximation algorithms with feasible computational complexity have been developed [1, 2, 11, 15]. Given a distance matrix that satisfies the 4-point condition, the associated unrooted tree that generated the matrix can be found in $\mathcal{O}(k^2)$ time, where $k$ is equal to the number of clusters [25].

We propose the following iterative algorithm to incorporate the 4-point condition into the optimization of Equation (1):

**Require:** $M \succeq 0$
**Ensure:** $(\Pi, Y) \approx (\Pi^*, Y^*)$ that solve Equation (1) with the constraints given in Equation (11)
   Initialize $Y = I$
   Initialize $\Pi$ using the relaxation in Section 3.1
   **while** Convergence has not been reached **do**
      Solve for $Y$ given $\Pi$ using Equation (8)
      Construct $D$ such that $D_{ij} = \sqrt{Y_{ii} + Y_{jj} - 2Y_{ij}}$
      Solve for $\min_{D_T} \|D - D_T\|^2$
      Assign $Y = -\frac{1}{2} H_k (D_T \odot D_T) H_k$, where $\odot$ represents the Hadamard product
      Update $\Pi$ using a normalized version of the algorithm described in [22]
   **end while**

One can view this optimization as solving the relaxed version of the problem such that $Y$ is only constrained to be positive definite, and then projecting the solution onto the feasible set by requiring $Y$ to be constructed from a tree metric. By iterating the procedure, we can allow $\Pi$ to reflect the fact that it should best fit the current estimate of the tree metric.

## 5 Experimental Results

To illustrate the effectiveness of the proposed algorithm, we have performed clustering on two benchmark datasets. The face dataset presented in [22] consists of 185 images of three different people, each with three different facial expressions. The authors posited that this would be best represented by a ternary tree structure, where the first level would decide which subject was represented, and the second level would be based on facial expression. In fact, their clustering algorithm roughly partitioned the data in this way when the appropriate structure matrix was imposed. We will show that our algorithm is able to find a similar structure without supervision, which better represents the empirical structure of the data.

We have also included results for the NIPS 1-12 dataset,[2] which consists of binarized histograms of the first 12 years of NIPS papers, with a vocabulary size of 13649 and a corpus size of 1740. A Gaussian kernel was used with the normalization parameter set to the median squared distance between points in input space.

### 5.1 Performance Evaluation on the Face Dataset

We first describe a numerical comparison on the face dataset [22] of the approach presented in Section 4 (where $M = H_n K H_n$ is assigned as in a HSIC objective). We considered two alternative approaches: a classic spectral clustering algorithm [21], and the dependence maximization approach of Song et al. [22]. Because the approach in [22] is not able to learn the structure of $Y$ from the data, we have optimized the partition matrix for 8 different plausible hierarchical structures (Figure 1). These have been constructed by truncating $n$-ary trees to the appropriate number of leaf nodes. For the evaluation, we have made use of the fact that the desired partition of the data is known for the face dataset, which allows us to compare the predicted clusters to the ground truth labels. For each

partition matrix, we compute the conditional entropy of the true labels, $l$, given the cluster ids, $c$, $H(l|c)$, which is related to mutual information by $I(l; c) = H(l) - H(l|c)$. As $H(l)$ is fixed for a given dataset, $\text{argmax}_c\, I(l; c) = \text{argmin}_c\, H(l|c)$, and $H(l|c) \geq 0$ with equality only in the case that the clusters are pure [9]. Table 1 shows the learned structure and proper normalization of our algorithm results in a partition of the images that much more closely matches the true identities and expressions of the faces, as evidenced by a much lower conditional entropy score than either the spectral clustering approach of [21] or the dependence maximization approach of [22].

Figure 2 shows the discovered taxonomy for the face dataset, where the length of the edges is proportional to the distance in the tree metric (thus, in interpreting the graph, it is important to take into account both the nodes at which particular clusters are connected, and the distance between these nodes; this is by contrast with Figure 1, which only gives the hierarchical cluster structure and does not represent distance). Our results show we have indeed recovered an appropriate tree structure without having to pre-specify the cluster similarity relations.

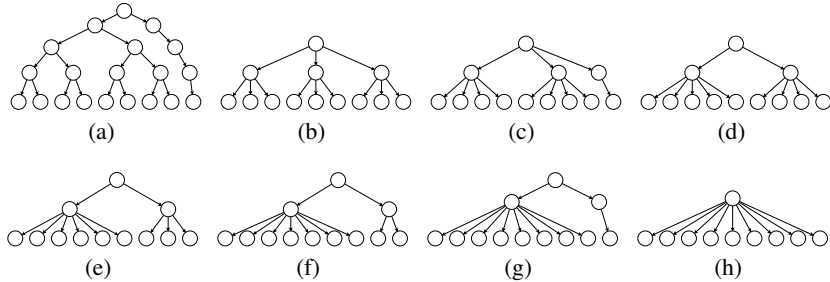

Figure 1: Structures used in the optimization of [22]. The clusters are identified with leaf nodes, and distances between the clusters are given by the minimum path length from one leaf to another. Each edge in the graph has equal cost.

| spectral | a | b | c | d | e | f | g | h | taxonomy |
|---|---|---|---|---|---|---|---|---|---|
| 0.5443 | 0.7936 | 0.4970 | 0.6336 | 0.8652 | 1.2246 | 1.1396 | 1.1325 | 0.5180 | **0.2807** |

Table 1: Conditional entropy scores for spectral clustering [21], the clustering algorithm of [22], and the method presented here (last column). The structures for columns a-h are shown in Figure 1, while the learned structure is shown in Figure 2. The structure for spectral clustering is implicitly equivalent to that in Figure 1(h), as is apparent from the analysis in Section 3.1. Our method exceeds the performance of [21] and [22] for all the structures.

## 5.2 NIPS Paper Dataset

For the NIPS dataset, we partitioned the documents into $k = 8$ clusters using the numerical taxonomy clustering algorithm. Results are given in Figure 3. To allow us to verify the clustering performance, we labeled each cluster using twenty informative words, as listed in Table 2. The most representative words were selected for a given cluster according to a heuristic score $\frac{\gamma}{\nu} - \frac{\eta}{\tau}$, where $\gamma$ is the number of times the word occurs in the cluster, $\eta$ is the number of times the word occurs outside the cluster, $\nu$ is the number of documents in the cluster, and $\tau$ is the number of documents outside the cluster. We observe that not only are the clusters themselves well defined (e.g cluster $a$ contains neuroscience papers, cluster $g$ covers discriminative learning, and cluster $h$ Bayesian learning), but the similarity structure is also reasonable: clusters $d$ and $e$, which respectively cover training and applications of neural networks, are considered close, but distant from $g$ and $h$; these are themselves distant from the neuroscience cluster at $a$ and the hardware papers in $b$; reinforcement learning gets a cluster at $f$ distant from the remaining topics. Only cluster $c$ appears to be indistinct, and shows no clear theme. Given its placement, we anticipate that it would merge with the remaining clusters for smaller $k$.

## 6 Conclusions and Future Work

We have introduced a new algorithm, numerical taxonomy clustering, for simultaneously clustering data and discovering a taxonomy that relates the clusters. The algorithm is based on a dependence

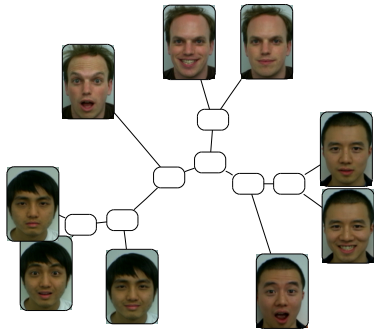
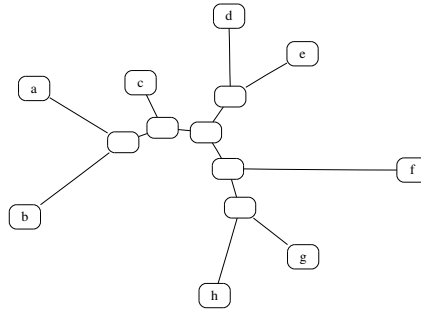

Figure 2: Face dataset and the resulting taxonomy that was discovered by the algorithm

Figure 3: The taxonomy discovered for the NIPS dataset. Words that represent the clusters are given in Table 2.

| a | b | c | d | e | f | g | h |
|---|---|---|---|---|---|---|---|
| neurons | chip | memory | network | training | state | function | data |
| cells | circuit | dynamics | units | recognition | learning | error | model |
| model | analog | image | learning | network | policy | algorithm | models |
| cell | voltage | neural | hidden | speech | action | functions | distribution |
| visual | current | hopfield | networks | set | reinforcement | learning | gaussian |
| neuron | figure | control | input | word | optimal | theorem | likelihood |
| activity | vlsi | system | training | performance | control | class | parameters |
| synaptic | neuron | inverse | output | neural | function | linear | algorithm |
| response | output | energy | unit | networks | time | examples | mixture |
| firing | circuits | capacity | weights | trained | states | case | em |
| cortex | synapse | object | error | classification | actions | training | bayesian |
| stimulus | motion | field | weight | layer | agent | vector | posterior |
| spike | pulse | motor | neural | input | algorithm | bound | probability |
| cortical | neural | computational | layer | system | reward | generalization | density |
| frequency | input | network | recurrent | features | sutton | set | variables |
| orientation | digital | images | net | test | goal | approximation | prior |
| motion | gate | subjects | time | classifier | dynamic | bounds | log |
| direction | cmos | model | back | classifiers | step | loss | approach |
| spatial | silicon | associative | propagation | feature | programming | algorithms | matrix |
| excitatory | implementation | attractor | number | image | rl | dimension | estimation |

Table 2: Representative words for the NIPS dataset clusters.

maximization approach, with the Hilbert-Schmidt Independence Criterion as our measure of dependence. We have shown several interesting theoretical results regarding dependence maximization clustering. First, we established the relationship between dependence maximization and spectral clustering. Second, we showed the optimal positive definite structure matrix takes the form of a set kernel, where sets are defined by cluster membership. This result applied to the original dependence maximization objective indicates that the inclusion of an unconstrained structure matrix does not affect the optimal partition matrix. In order to remedy this, we proposed to include constraints that guarantee $Y$ to be generated from an additive metric. Numerical taxonomy clustering allows us to optimize the constrained problem efficiently.

In our experiments on grouping facial expressions, numerical taxonomy clustering is more accurate than the existing approaches of spectral clustering and clustering with a fixed predefined structure. We were also able to fit a taxonomy to NIPS papers that resulted in a reasonable and interpretable clustering by subject matter. In both the facial expression and NIPS datasets, similar clusters are close together on the resulting tree.We conclude that numerical taxonomy clustering is a useful tool both for improving the accuracy of clusterings and for the visualization of complex data.

Our approach presently relies on the combinatorial optimization introduced in [22] in order to optimize $\Pi$ given a fixed estimate of $Y$. We believe that this step may be improved by relaxing the problem similar to Section 3.1. Likewise, automatic selection of the number of clusters is an interesting area of future work. We cannot expect to use the criterion in Equation (1) to select the number of clusters because increasing the size of $\Pi$ and $Y$ can never decrease the objective. However, the

elbow heuristic can be applied to the optimal value of Equation (1), which is closely related to the eigengap approach. Another interesting line of work is to consider optimizing a clustering objective derived from the Hilbert-Schmidt Normalized Independence Criterion (HSNIC) [13].

**Acknowledgments**

This work is funded by the EC projects CLASS, IST 027978, PerAct, EST 504321, and by the Pascal Network, IST 2002-506778. We would also like to thank Christoph Lampert for simplifying the Moore-Penrose generalized inverse.

## Footnotes

[1] For a detailed derivation, see the extended technical report [6].

[2]The NIPS 1-12 dataset is available at `http://www.cs.toronto.edu/~roweis/data.html`

## References

[1] R. Agarwala, V. Bafna, M. Farach, B. Narayanan, M. Paterson, and M. Thorup. On the approximability of numerical taxonomy (fitting distances by tree metrics). In *SODA*, pages 365–372, 1996.

[2] N. Ailon and M. Charikar. Fitting tree metrics: Hierarchical clustering and phylogeny. In *Foundations of Computer Science*, pages 73–82, 2005.

[3] F. R. Bach and M. I. Jordan. Learning spectral clustering, with application to speech separation. *JMLR*, 7:1963–2001, 2006.

[4] R. Baire. *Leçons sur les Fonctions Discontinues*. Gauthier Villars, 1905.

[5] C. Baker. Joint measures and cross-covariance operators. *Transactions of the American Mathematical Society*, 186:273–289, 1973.

[6] M. B. Blaschko and A. Gretton. Taxonomy inference using kernel dependence measures. Technical report, Max Planck Institute for Biological Cybernetics, 2008.

[7] D. Blei, T. Griffiths, M. Jordan, and J. Tenenbaum. Hierarchical topic models and the nested chinese restaurant process. In *NIPS 16*, 2004.

[8] P. Buneman. The Recovery of Trees from Measures of Dissimilarity. In D. Kendall and P. Tautu, editors, *Mathematics the the Archeological and Historical Sciences*, pages 387–395. Edinburgh U.P., 1971.

[9] T. M. Cover and J. A. Thomas. *Elements of Information Theory*. Wiley, 1991.

[10] N. Cristianini, J. Shawe-Taylor, A. Elisseeff, and J. Kandola. On kernel-target alignment. In *NIPS 14*, 2002.

[11] M. Farach, S. Kannan, and T. Warnow. A robust model for finding optimal evolutionary trees. In *STOC*, pages 137–145, 1993.

[12] K. Fukumizu, F. R. Bach, and M. I. Jordan. Dimensionality reduction for supervised learning with reproducing kernel Hilbert spaces. *JMLR*, 5:73–99, 2004.

[13] K. Fukumizu, A. Gretton, X. Sun, and B. Schölkopf. Kernel measures of conditional dependence. In *NIPS 20*, 2008.

[14] A. Gretton, O. Bousquet, A. Smola, and B. Schölkopf. Measuring statistical dependence with Hilbert-Schmidt norms. In *Algorithmic Learning Theory*, pages 63–78, 2005.

[15] B. Harb, S. Kannan, and A. McGregor. Approximating the best-fit tree under $l_p$ norms. In *APPROX-RANDOM*, pages 123–133, 2005.

[16] D. Haussler. Convolution kernels on discrete structures. Technical Report UCSC-CRL-99-10, University of California at Santa Cruz, 1999.

[17] R. A. Horn and C. R. Johnson. *Matrix Analysis*. Cambridge University Press, Cambridge, 1985.

[18] A. K. Jain and R. C. Dubes. *Algorithms for Clustering Data*. Prentice Hall, 1988.

[19] P. Macnaughton Smith, W. Williams, M. Dale, and L. Mockett. Dissimilarity analysis: a new technique of hierarchical subdivision. *Nature*, 202:1034–1035, 1965.

[20] C. D. Meyer, Jr. Generalized inversion of modified matrices. *SIAM Journal on Applied Mathematics*, 24(3):315–323, 1973.

[21] A. Y. Ng, M. I. Jordan, and Y. Weiss. On Spectral Clustering: Analysis and an Algorithm. In *NIPS*, pages 849–856, 2001.

[22] L. Song, A. Smola, A. Gretton, and K. M. Borgwardt. A Dependence Maximization View of Clustering. In *ICML*, pages 815–822, 2007.

[23] Y. W. Teh, M. I. Jordan, M. J. Beal, and D. M. Blei. Hierarchical dirichlet processes. *JASA*, 101(476):1566–1581, 2006.

[24] U. von Luxburg. A Tutorial on Spectral Clustering. *Statistics and Computing*, 17(4):395–416, 2007.

[25] M. S. Waterman, T. F. Smith, M. Singh, and W. A. Beyer. Additive Evolutionary Trees. *Journal of Theoretical Biology*, 64:199–213, 1977.

